# Thinning Measurement Models and Questionnaire Design

**Ricardo Silva**
Department of Statistical Science
University College London
Gower Street, London WC1E 6BT
`ricardo@stats.ucl.ac.uk`

## Abstract

Inferring key unobservable features of individuals is an important task in the applied sciences. In particular, an important source of data in fields such as marketing, social sciences and medicine is questionnaires: answers in such questionnaires are noisy measures of target unobserved features. While comprehensive surveys help to better estimate the latent variables of interest, aiming at a high number of questions comes at a price: refusal to participate in surveys can go up, as well as the rate of missing data; quality of answers can decline; costs associated with applying such questionnaires can also increase. In this paper, we cast the problem of refining existing models for questionnaire data as follows: solve a constrained optimization problem of preserving the maximum amount of information found in a latent variable model using only a subset of existing questions. The goal is to find an optimal subset of a given size. For that, we first define an information theoretical measure for quantifying the quality of a reduced questionnaire. Three different approximate inference methods are introduced to solve this problem. Comparisons against a simple but powerful heuristic are presented.

## 1 Contribution

A common goal in the applied sciences is to measure concepts of interest that are not directly observable (Bartholomew et al., 2008). Such is the case in the social sciences, medicine, economics and other fields, where quantifying key attributes such as "consumer satisfaction," "anxiety" and "recession" requires the development of *indicators*: observable variables that are postulated to measure the target latent variables up to some measurement error (Bollen, 1989; Carroll et al., 1995).

In a probabilistic framework, this often boils down to a latent variable model (Bishop, 1998). One common setup is to assume each observed indicator $Y_i$ as being generated independently given the set of latent variables $\mathbf{X}$. Conditioning on any given observed data point $\mathbf{Y}$ gives information about the distribution of the latent vector $\mathbf{X}$, which can then be used for ranking, clustering, visualization or smoothing, among other tasks. Figure 1 provides an illustration.

*Questionnaires* from large surveys are sometimes used to provide such indicators, each $Y_i$ recording an answer that typically corresponds to a Bernoulli or ordinal variable. For instance, experts can be given questions concerning whether there is freedom of press in a particular nation, as a way of measuring its democratization level (Bollen, 1989; Palomo et al., 2007). Nations can then be clustering or ranked within an interpretable latent space. Long questionnaires have nevertheless drawbacks, as summarized by Stanton et al. (2002) in the context of psychometric studies:

> Longer surveys take more time to complete, tend to have more missing data, and have higher refusal rates than short surveys. Arguably, then, techniques to reducing the length of scales while maintaining psychometric quality are worthwhile.

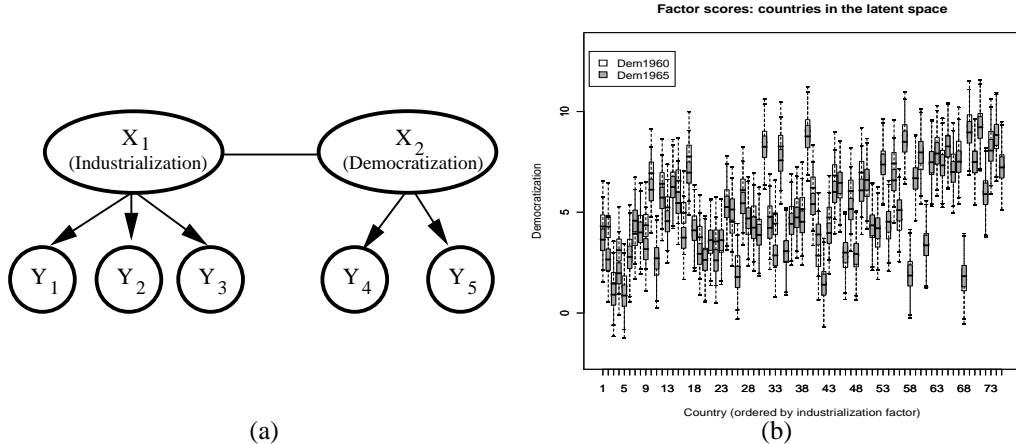

<div align="center">(a)</div>

<div align="center">(b)</div>

Figure 1: (a) A graphical representation of a latent variable model. Notice that in general latent variables will be dependent. Here, the question is how to quantify democratization and industrialization levels of nations given observed indicators $\mathbf{Y}$ such as freedom of press and gross national product, among others (Bollen, 1989; Palomo et al., 2007). (b) An example of a result implied by the model (adapted from Palomo et al. (2007)): barplots of the conditional distribution of democratization levels given the observed indicators at two time points, ordered by the posterior mean industrialization level. The distribution of the latent variables given the observations is the basis of the analysis.

Our contribution is a methodology for choosing which indicators to preserve (e.g., which items to keep in a questionnaire) given: i.) a latent variable model specification of the domain of interest; ii.) a target number of indicators that should be preserved. To accomplish this, we provide: i.) a target objective function that quantifies the amount of information preserved by a choice of a subset of indicators, with respect to the full set; ii.) algorithms for optimizing this choice of subset with respect to the objective function. The general idea is to start with a target posterior distribution of latent variables, defined by some latent variable measurement model $\mathcal{M}$ (i.e., $\mathcal{P}_{\mathcal{M}}(\mathbf{X} \mid \mathbf{Y})$). We want to choose a subset $\mathbf{Y_z} \subset \mathbf{Y}$ so that the resulting conditional distribution $\mathcal{P}_{\mathcal{M}}(\mathbf{X} \mid \mathbf{Y_z})$ is as close as possible to the original one according to some metric. Model $\mathcal{M}$ is provided either by expertise or by numerous standard approaches that can be applied to learn it from data (e.g., methods in Bishop, 2009). We call this task *measurement model thinning*.

Notice that the size of $\mathbf{Y_z}$ is a domain-dependent choice. Assuming $\mathcal{M}$ is a good model for the data, choosing a subset of indicators will incur some information loss. It is up to the analyst to choose a trade-off between loss of information and the design of simpler, cheaper ways of measuring latent variables. Even if a shorter questionnaire is not to be deployed, the outcome of measurement model thinning provides a formal sensitivity analysis of the target latent distribution with respect to the available indicators. The result is useful to generate different insights into the domain.

This paper is organized as follows: Section 2 defines a formal criterion to quantify how appropriate a subset $\mathbf{Y_z}$ is. Section 3 describes different approaches in which this criterion can be optimized. Related work is briefly discussed in Section 4. Experiments with synthetic and real data are discussed in Section 5, followed by the conclusion.

## 2    An Information-Theoretical Criterion

Our focus is on domains where latent variables are not a by-product of a dimensionality reduction technique, but the target of the analysis as in the example of Figure 1. That is, measurement error problems where the variables to be recorded are *designed* specifically to obtain information concerning such unknowns (Carroll et al., 1995; Bartholomew et al., 2008). As such, we postulate that the outcome of any analysis should be a functional of $\mathcal{P}_{\mathcal{M}}(\mathbf{X} \mid \mathbf{Y})$, the conditional distribution of unobservables $\mathbf{X}$ given observables $\mathbf{Y}$ within a model $\mathcal{M}$. It is assumed that $\mathcal{M}$ specifies the joint $\mathcal{P}_{\mathcal{M}}(\mathbf{X}, \mathbf{Y})$. We further assume that observed variables are conditionally independent given $\mathbf{X}$, i.e. $\mathcal{P}_{\mathcal{M}}(\mathbf{X}, \mathbf{Y}) = \mathcal{P}_{\mathcal{M}}(\mathbf{X}) \prod_{i=1}^{p} \mathcal{P}_{\mathcal{M}}(Y_i \mid \mathbf{X})$, with $p$ being the number of observed indicators.

If $\mathbf{z} \equiv (z_1, z_2, \ldots, z_p)$ is a binary vector of the same dimensionality as $\mathbf{Y}$, and $\mathbf{Y_z}$ is the subset of $\mathbf{Y}$ corresponding the non-zero entries of $\mathbf{z}$, we can assess $\mathbf{z}$ by the KL divergence

$$KL(\mathcal{P}_\mathcal{M}(\mathbf{X} \mid \mathbf{Y}) \,\|\, \mathcal{P}_\mathcal{M}(\mathbf{X} \mid \mathbf{Y_z})) \equiv \int \mathcal{P}_\mathcal{M}(\mathbf{X} \mid \mathbf{Y}) \log \frac{\mathcal{P}_\mathcal{M}(\mathbf{X} \mid \mathbf{Y})}{\mathcal{P}_\mathcal{M}(\mathbf{X} \mid \mathbf{Y_z})} \, d\mathbf{X}$$

This is well-defined, since both distributions lie in the same sample space despite the difference of dimensionality between $\mathbf{Y}$ and $\mathbf{Y_z}$. Moreover, since $\mathbf{Y}$ is itself a random vector, our criterion becomes the expected KL divergence

$$\langle KL(\mathcal{P}_\mathcal{M}(\mathbf{X} \mid \mathbf{Y}) \,\|\, \mathcal{P}_\mathcal{M}(\mathbf{X} \mid \mathbf{Y_z})) \rangle_{\mathcal{P}_\mathcal{M}(\mathbf{Y})}$$

where $\langle \cdot \rangle$ denotes expectation. Our goal is to minimize this function with respect to $\mathbf{z}$. Rearranging this expression to drop all constants that do not depend on $\mathbf{z}$, and multiplying it by $-1$ to get a maximization problem, we obtain the problem of finding $\mathbf{z}^\star$ such that

$$
\begin{aligned}
\mathbf{z}^\star &= argmax_\mathbf{z} \left\{ \langle \log(\mathcal{P}_\mathcal{M}(\mathbf{Y_z} \mid \mathbf{X})) \rangle_{\mathcal{P}_\mathcal{M}(\mathbf{X}, \mathbf{Y_z})} - \langle \log(\mathcal{P}_\mathcal{M}(\mathbf{Y_z})) \rangle_{\mathcal{P}_\mathcal{M}(\mathbf{Y_z})} \right\} \\
&= argmax_\mathbf{z} \left\{ \sum_{i=1}^{p} z_i \langle \log(\mathcal{P}_\mathcal{M}(Y_i \mid \mathbf{X})) \rangle_{\mathcal{P}_\mathcal{M}(\mathbf{X}, Y_i)} + \mathcal{H}_\mathcal{M}(\mathbf{Y_z}) \right\} \\
&\equiv argmax_\mathbf{z} \mathcal{F}_\mathcal{M}(\mathbf{z})
\end{aligned}
$$

subject to $\sum_{i=1}^{p} z_i = K$ for a choice of $K$, and $z_i \in \{0, 1\}$. $\mathcal{H}_\mathcal{M}(\cdot)$ denotes here the entropy of a distribution parameterized by $\mathcal{M}$. Notice we used the assumption that indicators are mutually independent given $\mathbf{X}$. There is an intuitive appeal of having a joint entropy term to reward not only marginal relationships between indicators and latent variables, but also selections that are jointly diverse. Notice that optimizing this objective function turns out to be equivalent to minimizing the conditional entropy of latent variables given $\mathbf{Y_z}$. Motivating conditional entropy from a more fundamental principle illustrates that other functions can be obtained by changing the divergence.

## 3 Approaches for Approximate Optimization

The problem of optimizing $\mathcal{F}_\mathcal{M}(\mathbf{z})$ subject to the constraints $\sum_{i=1}^{p} z_i = K$, $z_i \in \{0, 1\}$, is hard not only for its combinatorial nature, but due to the entropy term. This needs to be approximated, and the nature of the approximation should depend on the form taken by $\mathcal{M}$. We will assume that it is possible to efficiently compute any marginals of $\mathcal{P}_\mathcal{M}(\mathbf{Y})$ of modest dimensionality (say, 10 dimensions). This is the case, for instance, in the *probit* model for binary data:

$$\mathbf{X} \sim \mathcal{N}(\mathbf{0}, \Sigma), \qquad Y_i^\star \sim \mathcal{N}(\Lambda_i^\mathsf{T}\mathbf{X} + \lambda_{i;0}, 1),$$
$$Y_i = 1, \text{if } Y_i^\star > 0, \text{ and } 0 \text{ otherwise}$$

where $\mathcal{N}(\mathbf{m}, \mathbf{S})$ is the multivariate Gaussian distribution with mean $\mathbf{m}$ and covariance matrix $\mathbf{S}$. The probit model is one of the most common latent variable models for questionnaire data (Bartholomew et al., 2008), with a straigthforward extension to ordinal data. In this model, marginals for a few dozen variables can be obtained efficiently since this corresponds to calculating multivariate Gaussian probabilities (Genz, 1992). Parameters can be fit by a variety of methods (Hahn et al., 2010).

We also assume that $\mathcal{M}$ allows for the computation of $\langle \log(\mathcal{P}_\mathcal{M}(Y_i \mid \mathbf{X})) \rangle_{\mathcal{P}_\mathcal{M}(\mathbf{X}, Y_i)}$ at little cost. Again, in the binary probit model this is simple, since this requires integrating away a single binary variable $Y_i$ and a univariate Gaussian $\Lambda_i^\mathsf{T}\mathbf{X}$.

### 3.1 Gaussian Entropy

One approximation to $\mathcal{F}_\mathcal{M}(\mathbf{z})$ is to replace its entropy term by the corresponding entropy from some Gaussian distribution $P_\mathcal{N}(\mathbf{Y}_z)$. The entropy of a Gaussian distribution is proportional to the logarithm of the determinant of its covariance matrix, and hence can be computed in $\mathcal{O}(p^3)$ steps. This Gaussian can be chosen as the one closest to $P_\mathcal{M}(\mathbf{Y_z})$ in a $KL(P_\mathcal{M} \,\|\, P_\mathcal{N})$ sense: that is, the one with the same first and second moments as $P_\mathcal{M}(\mathbf{Y_z})$. In our case, computing these moments can be done deterministically (up to numerical error) using standard bivariate quadrature methods. No expectation-propagation (Minka, 2001) is necessary. The corresponding objective function is

$$\mathcal{F}_{\mathcal{M};\mathcal{N}}(\mathbf{z}) \equiv \sum_{i=1}^{p} z_i \langle \log(\mathcal{P}_\mathcal{M}(Y_i \mid \mathbf{X})) \rangle_{\mathcal{P}_\mathcal{M}(\mathbf{X}, Y_i)} + 0.5 \log |\Sigma_\mathbf{z}|$$

where $\Sigma_{\mathbf{z}}$ is the covariance matrix of $\mathbf{Y}_z$ – which for binary and ordinal data has a sensible interpretation. This function is also an upper bound on the exact function, $\mathcal{F}_{\mathcal{M}}(\mathbf{z})$, since the Gaussian is the distribution with the largest entropy for a given mean vector and covariance matrix. The resulting function is non-linear in $\mathbf{z}$. In our experiments, we optimize for $\mathbf{z}$ using a greedy scheme: for all possible pairs $(i, j)$ such that $z_i = 1$ and $z_j = 0$, we swap its values (so that $\sum_i z_i$ is always $K$). We choose the pair with the highest increase in $\mathcal{F}_{\mathcal{M};\mathcal{N}}(\mathbf{z})$ and repeat the process until convergence.

## 3.2  Entropy with Bounded Neighborhoods

An alternative bound can be derived from a standard fact in information theory: $\mathcal{H}(\mathbf{Y} \mid \mathcal{S}) \leq \mathcal{H}(\mathbf{Y} \mid \mathcal{S}')$ for $S' \subseteq S$, where $\mathcal{H}(\cdot \mid \cdot)$ denotes conditional entropy. This was exploited by Globerson and Jaakkola (2007) to define an upper bound in the entropy of a distribution as follows: consider a permutation $\mathbf{e}$ of the set $\{1, 2, \ldots, p\}$, with $e(i)$ being the $i$-th element of $\mathbf{e}$. Denote by $\mathbf{e}(1 : i)$ the first $i$ elements of this permutation (an empty set if $i < 1$). Moreover, let $N(\mathbf{e}, i)$ be a subset of $\mathbf{e}(1 : i - 1)$. For a given set variables $\mathbf{Y} = \{Y_1, Y_2, \ldots, Y_p\}$ the following bound holds:

$$\mathcal{H}(Y_1, Y_2, \ldots Y_p) = \sum_{i=1}^{n} \mathcal{H}(Y_{e(i)} \mid Y_{\mathbf{e}(1:i-1)}) \leq \sum_{i=1}^{p} \mathcal{H}(Y_{e(i)} \mid Y_{N(\mathbf{e},i)}) \qquad (1)$$

If each set $N(\mathbf{e}, i)$ is no larger than some constant $D$, then this bound can be computed in $\mathcal{O}(p \cdot 2^D)$ steps for binary probit models. The bound holds for any choice of $\mathbf{e}$, but we want it to be as tight as possible so that it gets weighted in a reasonable way against the other terms in $\mathcal{F}_{\mathcal{M}}(\cdot)$. Since the entropy function is decomposable as a sum of functions that depend on $i$ and $N(\mathbf{e}, i)$ only, one can minimize this bound with respect to $\mathbf{e}$ by using permutation optimization methods such as (Jaakkola et al., 2010). In our implementation, we use a method similar to Teyssier and Koller (2005) that shuffles neighboring entries of $\mathbf{e}$ to generate candidates, chooses the optimal $N(\mathbf{e}, i)$ for each $i$ given the candidate $\mathbf{e}$, and picks as the next permutation the candidate $\mathbf{e}$ with the greatest decrease in the bound.

Notice that a permutation choice $\mathbf{e}$ and neighborhood choices $N(\mathbf{e}, i)$ define a Bayesian network where $N(\mathbf{e}, i)$ are the parents of $Y_{e(i)}$. Therefore, if this Bayesian network model provides a good approximation to $\mathcal{P}_{\mathcal{M}}(\mathbf{Y})$, the bound will be reasonably tight.

Given $\mathbf{e}$, we will further relax this bound with the goal of obtaining an integer programming formulation for the problem of optimizing an upper bound to $\mathcal{F}_{\mathcal{M}}(\mathbf{z})$. For any given $\mathbf{z}$, we define the local term $\mathcal{H}_L(\mathbf{z}, i)$ as

$$\mathcal{H}_L(\mathbf{z}, i) \equiv \mathcal{H}_{\mathcal{M}}(Y_{e(i)} \mid \mathbf{Y}_{\mathbf{z}} \cap N(\mathbf{e}, i)) = \sum_{\mathcal{S} \in P(N(\mathbf{e},i))} \left[ \prod_{j \in \mathcal{S}} z_j \right] \left[ \prod_{k \in N(\mathbf{e},i) \setminus \mathcal{S}} (1 - z_k) \right] \mathcal{H}_{\mathcal{M}}(Y_{e(i)} \mid \mathcal{S})$$

$$(2)$$

where $P(\cdot)$ denotes the power set of a set. The new approximate objective function becomes

$$\mathcal{F}_{\mathcal{M};D}(\mathbf{z}) \equiv \sum_{i=1}^{p} z_i \langle \log(\mathcal{P}_{\mathcal{M}}(Y_i \mid \mathbf{X})) \rangle_{\mathcal{P}_{\mathcal{M}}(\mathbf{X},Y_i)} + \sum_{i=1}^{p} z_{e(i)} \mathcal{H}_L(\mathbf{z}, i) \qquad (3)$$

Notice that $\mathcal{H}_L(\mathbf{z}, i)$ is still an upper bound on $\mathcal{H}_{\mathcal{M}}(Y_{e(i)} \mid \mathbf{Y}_{\mathbf{e}(1:i-1)})$. The intuition is that we are bounding $\mathcal{H}_{\mathcal{M}}(\mathbf{Y}_{\mathbf{z}})$ by the entropy of a Bayesian network where a vertex $Y_{e(i)}$ is included if $z_{e(i)} = 1$, with corresponding parents given by $\mathbf{Y}_{\mathbf{z}} \cap N(\mathbf{e}, i)$. This is a well-defined Bayesian network for any choice of $\mathbf{z}$. The shortcoming is that ideally we would like this Bayesian network to be the actual marginal of the model given by $\mathbf{e}$ and $N(\mathbf{e}, i)$. It is not: if the network implied by $\mathbf{e}$ and $N(\mathbf{e}, i)$ was, for instance, $Y_1 \to Y_2 \to Y_3$, the choice of $\mathbf{z} = (1, 0, 1)$ would result on the entropy of the disconnected graph $\{Y_1, Y_3\}$, while the true marginal would correspond instead to the graph $Y_1 \to Y_3$. However, our simplified marginalization has the advantage of avoiding an intractable problem. Moreover, it allows us to redefine the problem as an integer linear program (ILP).

Each product $z_{e(i)} \prod_j z_j \prod_k (1 - z_k)$ appearing in (3) results in a sum of $\mathcal{O}(2^D)$ terms, each of which has (up to a sign) the form $q_M \equiv \prod_{m \in M} z_m$ for some set $M$. It is still the case that $q_M \in \{0, 1\}$. Therefore, objective function (3) can be interpreted as being linear on a set of binary variables $\{\{z\}, \{q\}\}$. We need further to enforce the constraints coming from

$$q_M = 1 \Rightarrow \{\forall m \in M, z_m = 1\}; \quad q_M = 0 \Rightarrow \{\exists m \in M \text{ s.t. } z_m = 0\}$$

It is well-known (Glover and Woolsey, 1974) that this corresponds to the linear constraints

$$
\begin{array}{lll}
q_M = 1 \Rightarrow \{\forall m \in M, z_m = 1\} & \Leftrightarrow & \forall m \in M, q_M - z_m \le 0 \\
q_M = 0 \Rightarrow \{\exists m \in M \text{ s.t. } z_m = 0\} & \Leftrightarrow & \sum_{m \in M} z_m - q_M \le |M| - 1
\end{array}
$$

which combined with the linear constraint $\sum_{i=1}^{p} z_i = K$ implies that optimizing $\mathcal{F}_{\mathcal{M};D}(\mathbf{z})$ is an ILP with $\mathcal{O}(p \cdot 2^D)$ variables and $\mathcal{O}(p^2 \cdot 2^D)$ constraints. In our experiments in Section 5, we were able to solve essentially all of such ILPs exactly using linear programming relaxations with branch-and-bound.

## 3.3 Entropy with Tree-Structured Bounds

The previous bound simplifies marginalization, which might badly overestimate entropies where the corresponding $\mathbf{Y_z}$ are uniformly spread out in permutation $\mathbf{e}$. We now propose a different type of bound which treats different marginalizations on an equal footing. It comes from the following observation: since $\mathcal{H}(Y_{e(i)} \mid \mathbf{Y}_{\mathbf{e}(1:i-1)})$ is less than or equal to any conditional entropy $\mathcal{H}(Y_{e(i)} \mid Y_j)$ for $j \in \mathbf{e}(1 : i - 1)$, we have that the tightest bound given by singleton conditioning sets is

$$
\mathcal{H}(Y_{e(i)} \mid \mathbf{Y}_{\mathbf{e}(1:i-1)}) \le \min_{j \in \mathbf{e}(1:i-1)} \mathcal{H}_{\mathcal{M}}(Y_{e(i)} \mid Y_j),
$$

resulting in the objective function

$$
\mathcal{F}_{\mathcal{M};tree}(\mathbf{z}) \equiv \sum_{i=1}^{p} z_i \, \langle \log(\mathcal{P}_{\mathcal{M}}(Y_i \mid \mathbf{X})) \rangle_{\mathcal{P}_{\mathcal{M}}(\mathbf{X}, Y_i)} + \sum_{i=1}^{p} z_{e(i)} \cdot \min_{\{Y_j \in \mathbf{Y}_{\mathbf{e}(1:i-1)} \cap \mathbf{Y_z}\}} \mathcal{H}(Y_{e(i)} \mid Y_j) \quad (4)
$$

where $\min_{\{Y_j \in \mathbf{Y}_{\mathbf{e}(1:i-1)} \cap \mathbf{Y_z}\}} \mathcal{H}(Y_{e(i)} \mid Y_j) \equiv \mathcal{H}(Y_{e(i)})$ if $\mathbf{Y}_{\mathbf{e}(1:i-1)} \cap \mathbf{Y_z} = \emptyset$. The intuition is that we are bounding the exact entropy using the entropy of a directed tree rooted at $Y_{e_z(1)}$, the first element of $\mathbf{Y_z}$ according to $\mathbf{e}$. That is, all variables are marginally dependent in the approximation regardless of what $\mathbf{z}$ is, and for a fixed $\mathbf{z}$ the tree is, by construction, the one obtained by the usual greedy algorithm of adding edges corresponding to the next legal pair of vertices with maximum mutual information (following an ordering, in this case).

It turns out we can also write (4) as a linear objective function of a polynomial number of $0\backslash 1$ variables and constraints. Let $\bar{z}_i \equiv 1 - z_i$. Let $H_i^{(1)}, H_i^{(2)}, \ldots, H_i^{(i-1)}$ be the values of set $\{\mathcal{H}_{\mathcal{M}}(Y_{e(i)} \mid Y_{e(1)}), \ldots, \mathcal{H}_{\mathcal{M}}(Y_{e(i)} \mid Y_{e(i-1)})\}$ sorted in ascending order, with $z_i^{(1)}, \ldots, z_i^{(i-1)}$ being the corresponding permutation of $\{z_{e(1)}, \ldots, z_{e(i-1)}\}$. We have

$$
\begin{aligned}
\min_{\{Y_j \in \mathbf{Y}_{\mathbf{e}(1:i-1)} \cap \mathbf{Y_z}\}} \mathcal{H}(Y_{e(i)} \mid Y_j) &= z_i^{(1)} H_i^{(1)} + \bar{z}_i^{(1)} z_i^{(2)} H_i^{(2)} + \bar{z}_i^{(1)} \bar{z}_i^{(2)} z_i^{(3)} H_i^{(3)} + \ldots \\
& \quad \bar{z}_i^{(1)} \ldots \bar{z}_i^{(i-2)} z_i^{(i-1)} H_i^{(i-1)} + \prod_{j=1}^{i-1} \bar{z}_i^{(j)} \mathcal{H}_{\mathcal{M}}(Y_{e(i)}) \\
&\equiv \sum_{j=1}^{i-1} q_i^{(j)} H_i^{(j)} + q_i^{(i)} \mathcal{H}_{\mathcal{M}}(Y_{e(i)})
\end{aligned}
$$

where $q_i^{(j)} \equiv z_i^{(j)} \prod_{k=1}^{j-1} \bar{z}_i^{(k)}$, and also a binary $0\backslash 1$ variable. Plugging this expression into (4) gives a linear objective function in this extended variable space. The corresponding constraints are

$$
\begin{aligned}
q_i^{(j)} = 1 &\Rightarrow \{\forall z_m \in \{\bar{z}_i^{(1)}, \ldots, \bar{z}_i^{(j-1)}, z_i^{(j)}\}, z_m = 1\} \\
q_i^{(j)} = 0 &\Rightarrow \{\exists z_m \in \{\bar{z}_i^{(1)}, \ldots, \bar{z}_i^{(j-1)}, z_i^{(j)}\} \text{ s.t. } z_m = 0\}
\end{aligned}
$$

which, as shown in the previous section, can be written as linear constraints (substituting each $\bar{z}_i$ by $1 - z_i$). The total number of constraints is however $\mathcal{O}(p^3)$, which can be expensive, and often a linear relaxation procedure with branch-and-bound fails to provide guarantees of optimality.

## 3.4 The Reliability Score

Finally, it is important to design cheap, effective criteria whose maxima correlate with the maxima of $\mathcal{F}_{\mathcal{M}}(\cdot)$. Empirically, we have found high quality selections in binary probit models using the solution to the problem

$$
\text{maximize } \mathcal{F}_{\mathcal{M};\mathcal{R}}(\mathbf{z}) = \sum_{i=1}^{p} w_i z_i, \text{ subject to } z_i \in \{0, 1\}, \sum_{i=1}^{p} z_i = K
$$

where $w_i = \Lambda_i^\mathsf{T} \Sigma \Lambda_i$. This can be solved by picking the corresponding indicators with the highest $K$ weights $w_i$. Assuming a probit model where the measurement error for each $Y_i^\star$ has the same variance of 1, this score is related to the "reliability" of an indicator. Simply put, the reliability $R_i$ of an indicator is the proportion of its variance that is due to the latent variables (Bollen, 1989, Chapter 6): $R_i = w_i/(w_i + 1)$ for each $Y_i^\star$. There is no current theory linking this solution to the problem of maximizing $\mathcal{F}_\mathcal{M}(\cdot)$: since there is no entropy term, we can set an adversarial problem to easily defeat this method. For instance, this happens in a model where the $K$ indicators of highest reliability all measure the same latent variable $X_i$ and nothing else – much information about $X_i$ would be preserved, but little about other variables. In any case, we found this criterion to be fairly competitive even if at times it produces extreme failures. An honest account of more sophisticated selection mechanisms cannot be performed without including it, as we do in Section 5.

## 4   Related Work

The literature on survey analysis, in the context of latent variable models, contains several examples of guidelines on how to simplify questionnaires (sometimes described as providing "shortened versions" of scales). Much of the literature, however, consists of describing general guidelines and rules-of-thumb to accomplish this task (e.g, Richins, 2004; Stanton et al., 2002). One possible exception is Leite et al. (2008), which uses different model fitness criteria with respect to a given dataset to score candidate solutions, along with an expensive combinatorial optimization method. This conflates model selection and questionnaire thinning, and there is no theory linking the score functions to the amount of information preserved. In the machine learning and statistics literature, there is a large body of research in active learning, which is related to our task. One of the closest approaches is the one by Liang et al. (2009), which casts the classical problem of measurement selection within a Bayesian graphical model perspective. In that work, one has to choose which measurements to add. This is done sequentially, partially motivated by problems where collecting new measurements can be done relatively fast and cheap (say, by paying graduate students to annotate text data), and so the choice of next measurement can make use of fresh data. In our case, it not might be realistic to expect we can perform a large number of iterations of data collection – and as such the task of reducing the number of measurements from a large initial collection might be more relevant in practice. Liang et al. also focus on (multivariate) supervised learning instead of purely unsupervised learning. In statistics there is also a considerable body of literature on sufficient dimension reduction and its sparse variants (e.g., Chen et al., 2010). Such techniques create a bottleneck between two sets of variables in a regression problem (say, the mapping from $\mathbf{Y}$ to $\mathbf{X}$) while eliminating some of the input variables. In principle one might want to adapt such models to take a latent variable model $\mathcal{M}$ as the target mapping. Besides some loss of interpretability, the computational implications might be problematic, though. Moreover, this framework has another free parameter corresponding to the dimensionality of the bottleneck that has to be set. It is not clear how this parameter, along with a choice of sparsity level, would interact with a fixed choice $K$ of indicators to be kept.

## 5   Experiments

In this section, we first describe some synthetic experiments to provide insights about the different methods, followed by one brief description of a case study. In all of the experiments, the target models $\mathcal{M}$ are binary probit. We set the neighborhood parameter for $\mathcal{F}_{\mathcal{M}:\mathcal{N}}(\cdot)$ to 9. The ordering $\mathbf{e}$ for the tree-structured method is obtained by the same greedy search of Section 3.2, where now the score is the average of all $\mathcal{H}(Y_i \mid Y_j)$ for all $Y_j$ preceding $Y_i$. Finally, all ordering optimization methods were initialized by sorting indicators in a descending order according to their reliability scores, and the initial solution for all entropy-based optimization methods was given by the reliability score solution of Section 3.4. The integer program solver GUROBI 4.02 was used in all experiments.

### 5.1   Synthetic studies

We start with a batch of synthetic experiments. We generated 80 models with 40 indicators and 10 latent variables[1]. We further preprocess such models into two groups: in 40 of them, we select a

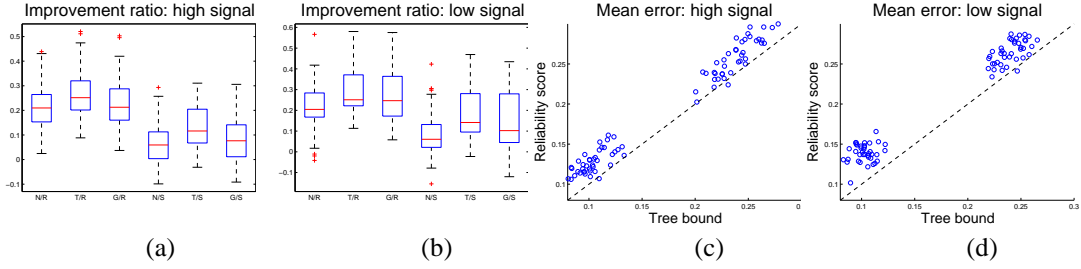

(a)          (b)          (c)          (d)

Figure 2: (a) A comparison of the bounded neighborhood ($N$), tree-based ($T$) and Gaussian ($G$) methods with respect to a random solution ($R$) and the reliability score ($S$). (b) A similar comparison for models where indicators are more weakly correlated to the latent variables than in (a). (c) and (d) Scatterplots of the average absolute deviance for the tree-based method (horizontal axis) against the reliability method (vertical axis). The bottom-left clouds correspond to the $K = 32$ trials.

target reliability $r_i$ for each indicator $Y_i$, uniformly at random from the interval [0.4 0.7]. We then rescale coefficients $\Lambda_i$ such that the reliability (defined in Section 3.4) of the respective $Y_i^\star$ becomes $r_i$. For the remaining 40 models, we sample $r_i$ uniformly at random from the interval [0.2 0.4].

We perform two choices of subsets: sets $\mathbf{Y}_z$ of size 20 and 32 (50% and 80% of the total number of indicators). Our evaluation is as follows: since the expected value is perhaps the most common functional of the posterior distribution $\mathcal{P}_\mathcal{M}(\mathbf{X} \mid \mathbf{Y})$, we calculate the expected value of the latent variables for a sample $\{\mathbf{y}^{(1)}, \mathbf{y}^{(2)}, \ldots, \mathbf{y}^{(1000)}\}$ of size 1000 taken from the respective synthetic models. This is done for the full set of 40 indicators, and for each set chosen by our four criteria: for each data point $i$ and each objective function $\mathcal{F}$, we evaluate the average distance $d_\mathcal{F}^{(i)} \equiv \sum_{j=1}^{10} |\hat{x}_j^{(i)} - \hat{x}_{j;\mathcal{F}}^{(i)}|/10$. In this case, $\hat{x}_j^{(i)}$ is the expected value of $X_j$ obtained by conditioning on all indicators, while $\hat{x}_{j;\mathcal{F}}^{(i)}$ is the one obtained with the subset selected by optimizing $\mathcal{F}$. We denote by $m_\mathcal{F}$ the average of $\{d_\mathcal{F}^{(1)}, d_\mathcal{F}^{(2)}, \ldots, d_\mathcal{F}^{(1000)}\}$. Finally, we compare the three main methods with respect to the reliability score method using the improvement ratio statistic $s_\mathcal{F} = 1 - m_\mathcal{F}/m_{\mathcal{F}_{\mathcal{M};\mathcal{R}}}$, the proportion of average error decrease with respect to the reliability score. In order to provide a sense of scale on the difficulty of each problem, we compute the same ratios with respect to a random selection, obtained by choosing $K = 20$ and $K = 32$ indicators uniformly at random.

Figure 2 provides a summary of the results. In Figure 2(a), each boxplot shows the distribution over the 40 probit models where reliabilities were sampled between [0.4 0.7] (the "high signal" models). The first three boxplots show the scores $s_\mathcal{F}$ of the bounded neighborhood, tree-structured and Gaussian methods, respectively, compared against random selections. The last three boxplots are comparisons against the reliability heuristic. The tree-based method easily beats the Gaussian method, with about 75% of its outcomes being better than the median Gaussian outcome. The Gaussian approach is also less reliable, with results showing a long lower tail. Although the reliability score is on average a good approach, in only a handful of cases it was better than the tree-based method, and by considerably smaller magnitudes compared to the upper tails in the tree-based outcome distribution. A separate panel (Figure 2(b)) is shown for the 40 models with lower reliabilities. In this case, all methods show stronger improvements over the reliability score, although now there is a less clear difference between the tree method and the Gaussian one. Finally, in panels (c) and (d) we present scatterplots for the average deviances $m_\mathcal{F}$ of the tree-based method against the reliability score. The two clouds correspond to the solutions with 20 and 32 indicators. Notice that in the vast majority of the cases the tree-based method does better.

---

We then rescale the matrix to make all variances equal to 1. We also generate 40 models using as the inverse Wishart scale matrix the correlation matrix will all off-diagonal entries set to 0.5. Coefficients linking indicators to latent variables were set to zero with probability 0.8, and sampled from a standard Gaussian otherwise. If some latent variable ends up with no child, or an indicator ends up with no parent, we uniformly choose one child/parent to be linked to it. Code to fully replicate the synthetic experiments is available at HTTP://WWW.HOMEPAGES.UCL.AC.UK/~UCGTRBD/.

## 5.2 Case study

The National Health Service (NHS) is the public health system of the United Kingdom. In 2009, a major survey called the *National Health Service National Staff Survey* was deployed with the goal of "collect(ing) staff views about working in their local NHS trust" (Care Quality Comission and Aston University, 2010). A questionnaire of 206 items was filled by $156,951$ respondents. We designed a measurement model based on the structure of the questionnaire and fit it by the posterior expected value estimator. Gaussian and inverse Wishart priors were used, along with Gibbs sampling and a random subset of $50,000$ respondents. See the Supplementary Material for more details. Several items in this survey asked for the NHS staff member to provide degrees of agreement in a Likert scale (Bartholomew et al., 2008) to questions such as

- ... have you ever come to work despite not feeling well enough to perform ... ?
- Have you felt pressure from your manager to come to work?
- Have you felt pressure from colleagues to come to work?
- Have you put yourself under pressure to come to work?

as different probes into an unobservable self-assessed level of work pressure.

We preprocessed and binarized the data to first narrow it down to 63 questions. We compare selections of 32 (50%) and 50 (80%) items using the same statistics of the previous section.

|  | $s_{\mathcal{F};D}$ | $s_{\mathcal{F};tree}$ | $s_{\mathcal{F};N}$ | $s_{\mathcal{F};random}$ | $m_{\mathcal{F};tree}$ | $m_{\mathcal{F};R}$ |
|---|---|---|---|---|---|---|
| $K = 32$ | 7.8% | 6.3% | 0.01% | $-16.0\%$ | 0.238 | 0.255 |
| $K = 50$ | 10.5% | 11.9% | 7.6% | $-0.05\%$ | 0.123 | 0.140 |

Although gains were relatively small (as measured by the difference between reconstruction errors $m_{\mathcal{F};tree} - m_{\mathcal{F};R}$ and the good performance of a random selection), we showed that: i.) we do improve results over a popular measure of indicator quality; ii.) we do provide some guarantees about the diversity of the selected items via a information-theoretical measure with an entropy term, with theoretically sound approximations to such a measure. For more details on the preprocessing, and more insights into the different selections, please refer to the Supplementary Material.

## 6 Conclusion

There are problems where one posits that the relevant information is encoded in the posterior distribution of a set of latent variables. Questionnaires (and other instruments) can be used as evidence to generate this posterior, but there is a cost associated with complex questionnaires. One problem is how to simplify such instruments of measurement. To the best of our knowledge, we provide the first formal account on how to solve it. Nevertheless, we would like to stress there is no substitute for common sense. While the tools we provide here can be used for a variety of analyses – from deploying simpler questionnaires to sensitivity analysis – the value and cost of keeping particular indicators can go much beyond the information contained in the latent posterior distribution. How to combine this criterion with other domain-dependent criteria is a matter of future research.

Another problem of importance is how to deal with model specification and transportability across studies. A measurement model built for a very specific population of respondents might transfer poorly to another group, and therefore taking into account model uncertainty will be important. The Bayesian setup discussed by Liang et al. (2009) might provide some directions on this issue. Also, there is further structure in real-world questionnaires we are not exploiting in the current work. Namely, it is not uncommon to have questionnaires with branching questions and other dynamic behaviour more commonly associated with Web based surveys and/or longitudinal studies. Finally, hybrid approaches combining the bounded neighborhood and the tree-structured methods, along with more sophisticated ordering optimization procedures and the use of other divergence measures and determinant-based criteria (e.g. Kulesza and Taskar, 2011), will also be studied in the future.

### Acknowledgments

The author would like to thank James Cussens and Simon Lacoste-Julien for helpful discussions, as well as the anonymous reviewers for further comments.

## Footnotes

[1]Details on the model generation: we generate 40 models by sampling the latent covariance matrix from an inverse Wishart distribution with 10 degrees of freedom and scale matrix $10\mathbf{I}$, $\mathbf{I}$ being the identity matrix.

# References

D. Bartholomew, F. Steele, I. Moustaki, and J. Galbraith. *Analysis of Multivariate Social Science Data, 2nd edition*. Chapman & Hall, 2008.

C. Bishop. Latent variable models. *In M. Jordan (editor), Learning in Graphical Models*, pages 371–403, 1998.

C. Bishop. *Pattern Recognition and Machine Learning*. Springer, 2009.

K. Bollen. *Structural Equations with Latent Variables*. John Wiley & Sons, 1989.

R. Carroll, D. Ruppert, and L. Stefanski. *Measurement Error in Nonlinear Models*. Chapman & Hall, 1995.

X. Chen, C. Zou, and R. Cook. Coordinate-independent sparse sufficient dimension reduction and variable selection. *Annals of Statistics*, 38:3696–3723, 2010.

Care Quality Commission and Aston University. Aston Business School, National Health Service National Staff Survey, 2009 [computer file]. *Colchester, Essex: UK Data Archive [distributor], October 2010. Available at* HTTPS://WWW.ESDS.AC.UK*, SN: 6570*, 2010.

A. Genz. Numerical computation of multivariate normal probabilities. *Journal of Computational and Graphical Statistics*, 1:141–149, 1992.

A. Globerson and T. Jaakkola. Approximate inference using conditional entropy decompositions. *Proceedings of the 11th International Conference on Artificial Intelligence and Statistics (AIS-TATS 2007)*, pages 141–149, 2007.

F. Glover and E. Woolsey. Converting the 0-1 polynomial programming problem to a 0-1 linear program. *Operations Research*, 22:180–182, 1974.

P. Hahn, J. Scott, and C. Carvalho. A sparse factor-analytic probit model for congressional voting patterns. *Duke University Department of Statistical Science, Discussion Paper 2009-22*, 2010.

T. Jaakkola, D. Sontag, A. Globerson, and M. Meila. Learning Bayesian network structure using LP relaxations. *Proceedings of the 13th International Conference on Artificial Intelligence and Statistics (AISTATS 2010)*, pages 366–373, 2010.

A. Kulesza and B. Taskar. k-DPPs: fixed-size determinantal point processes. *Proceedings of the 28th International Conference on Machine Learning (ICML)*, pages 1193–1200, 2011.

W. Leite, I-C. Huang, and G. Marcoulides. Item selection for the development of short forms of scales using an ant colony optimization algorithm. *Multivariate Behavioral Research*, 43:411–431, 2008.

P. Liang, M. Jordan, and D. Klein. Learning from measurements in exponential families. *Proceedings of the 26th Annual International Conference on Machine Learning (ICML '09)*, 2009.

T. Minka. A family of algorithms for approximate Bayesian inference. *PhD Thesis, Massachussets Institute of Technology*, 2001.

J. Palomo, D. Dunson, and K. Bollen. Bayesian structural equation modeling. *In Sik-Yum Lee (ed.), Handbook of Latent Variable and Related Models*, pages 163–188, 2007.

M. Richins. The material values scale: Measurement properties and development of a short form. *The Journal of Consumer Research*, 31:209–219, 2004.

J. Stanton, E. Sinar, W. Balzer, and P. Smith. Issues and strategies for reducing the length of self-reported scales. *Personnel Psychology*, 55:167–194, 2002.

M. Teyssier and D. Koller. Ordering-based search: A simple and effective algorithm for learning Bayesian networks. *Proceedings of the Twenty-first Conference on Uncertainty in AI (UAI '05)*, pages 584–590, 2005.

